# Representing Face Images for Emotion Classification

**Curtis Padgett**
Department of Computer Science
University of California, San Diego
La Jolla, CA 92034

**Garrison Cottrell**
Department of Computer Science
University of California, San Diego
La Jolla, CA 92034

## Abstract

We compare the generalization performance of three distinct representation schemes for facial emotions using a single classification strategy (neural network). The face images presented to the classifiers are represented as: full face projections of the dataset onto their eigenvectors (eigenfaces); a similar projection constrained to eye and mouth areas (eigenfeatures); and finally a projection of the eye and mouth areas onto the eigenvectors obtained from 32x32 random image patches from the dataset. The latter system achieves 86% generalization on novel face images (individuals the networks were not trained on) drawn from a database in which human subjects consistently identify a single emotion for the face.

## 1 Introduction

Some of the most successful research in machine perception of complex natural image objects (like faces), has relied heavily on reduction strategies that encode an object as a set of values that span the principal component sub-space of the object's images [Cottrell and Metcalfe, 1991, Pentland et al., 1994]. This approach has gained wide acceptance for its success in classification, for the efficiency in which the eigenvectors can be calculated, and because the technique permits an implementation that is biologically plausible. The procedure followed in generating these face representations requires normalizing a large set of face views ("mug-shots") and from these, identifying a statistically relevant sub-space. Typically the sub-space is located by finding either the eigenvectors of the faces [Pentland et al., 1994] or the weights of the connections in a neural network [Cottrell and Metcalfe, 1991].

In this work, we classify face images based on their emotional content and examine how various representational strategies impact the generalization results of a classifier. Previous work using whole face representations for emotion classification by

Cottrell and Metcalfe [Cottrell and Metcalfe, 1991] was less encouraging than results obtained for face recognition. We seek to determine if the problem in Cottrell and Metcalfe's work stems from bad data (i.e., the inability of the undergraduates to demonstrate emotion), or an inadequate representation (i.e. eigenfaces).

Three distinct representations of faces are considered in this work- a whole face representation similar to that used in previous work on recognition, sex, and emotion [Cottrell and Metcalfe, 1991]; a more localized representation based on the eyes (eigeneyes and eigenmouths) and mouth [Pentland et al., 1994]; and a representation of the eyes and mouth that makes use of basis vectors obtained by principal components of random image blocks. By examining the generalization rate of the classifiers for these different face representations, we attempt to ascertain the sensitivity of the representation and its potential for broader use in other vision classification problems.

## 2 Face Data

The dataset used in Cottrell and Metcalfe's work on emotions consisted of the faces of undergraduates who were asked to pose for particular expressions. However, feigned emotions by untrained individuals exhibit significant differences from the prototypical face expression [Ekman and Friesen, 1977]. These differences often result in disagreement between the observed emotion and the expression the actor is attempting to feign. A feigned smile for instance, differs around the eyes when compared with a "natural" smile. The quality of the displayed emotion is one of the reasons cited by Cottrell and Metcalfe for the poor recognition rates achieved by their classifier.

To reduce this possibility, we made use of a validated facial emotion database (Pictures of Facial Affect) assembled by Ekman and Friesen [Ekman and Friesen, 1976]. Each of the face images in this set exhibits a substantial agreement between the labeled emotion and the observed response of human subjects. The actors used in this database were trained to reliably produce emotions using Facial Action Coding System [Ekman and Friesen, 1977] and their images were presented to undergraduates for testing. The agreement between the emotion the actor was required to express and the students' observations was at least 70% on all the images incorporated in the database. We digitized a total of 97 images from 12 individuals (6 male, 6 female). Each portrays one of 7 emotions- happy, sad, fear, anger, surprise, disgust or neutral. With the exception of the neutral faces, each image in the set is labeled with a response vector of the remaining six emotions indicating the fraction of total respondents classifying the image with a particular emotion.

Each of the images was linearly stretched over the 8 bit greyscale range to reduce lighting variations. Although care was taken in collecting the original images, natural variations in head size and the mouth's expression resulted in significant variation in the distance between the eyes (2.7 pixels) and in the vertical distance from the eyes to the mouth (5.0 pixels). To achieve scale invariance, each image was scaled so that prominent facial features were located in the same image region. Eye and mouth templates were constructed from a number of images, and the most correlated template was used to localize the respective feature. Similar techniques have been employed in previous work on faces [Brunelli and Poggio, 1993]. Examples of the normalized images and typical facial expressions can be found in Figure 1.

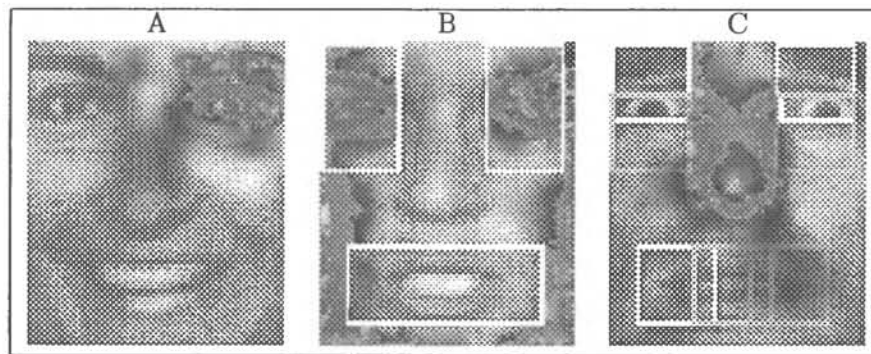

Figure 1: The image regions from which the representations are derived. Image A is a typical normalized and cropped image used to generate the full face eigenvectors. Image B depicts the feature regions from which the feature eigenvectors are calculated. Image C indicates each of the block areas projected onto the random block eigenvectors.

## 3   Representation

From the normalized database, we develop three distinct representations that form independent pattern sets for a single classification scheme. The selected representations differ in their scope (features or whole face) and in the nature of the sub-space (eigen- faces/features or eigenvectors of random image patches). The more familiar representational schemes (eigenfaces, eigenfeatures) are based on PCA of aligned features or faces. They have been shown to provide a reasonably compact representation for recognition purposes but little is known about their suitability for other classification tasks.

Random image patches are used to identify an alternative sub-space from which a set of localized face feature patterns are generated. This space is different in that the sub-space is more general, the variance captured by the leading eigenvectors is derived from patches drawn randomly over the set of face images. As we seek to develop generalizations across the rather small portion of image space containing faces or features, perturbations in this space will hopefully reflect more about class characteristics than individual distinctions.

For each of the pattern sets, we normalized the resultant set of values obtained from their projections on the eigenvectors by their standard deviation to produce Z scores. The Z score obtained from each image constitutes a single input to the neural network classifier. The highest valued eigenvectors typically contain more average features so that presumably they would be more suitable for object classification. All the representations will make use of the top $k$ principal components.

The full-faced pattern has proved to be quite useful in identification and the same techniques using face features have also been valuable [Pentland et al., 1994, Cottrell and Metcalfe, 1991]. However representations useful for identification of individuals may not be suitable for emotion recognition. In determining the appropriate emotion, structural differences in faces need to be suppressed. One way to accomplish this is to eliminate portions of the face image where variation provides little information with respect to emotion. Local changes in facial muscles around the eyes and mouth are generally associated with our perception of emotions [Ekman and Friesen, 1977]. The full face images presumably contain much information that is simply irrelevant to the task at hand which could impact the ability of the classifier to uncover the signal.

The feature based representations are derived from local windows around the eyes and mouth of the normalized whole face images (see Fig. 1B). The eigenvectors of the feature sub-space are determined independently for each feature (left/right eye and mouth). A face pattern is generated by projecting the particular facial features on their respective eigenvectors.

The random block pattern set is formed from image blocks extracted around the feature locations (see Fig. 1C). The areas around each eye are divided into two vertically overlapping blocks of size 32x32 and the mouth is sectioned into three. However, instead of performing PCA on each individual block or all of them together, a more general PCA of random 32x32 blocks taken over the entire image was used to generate the eigenvectors. We used random blocks to reduce the uniqueness of a projection for a single individual and provide a more reasonable model of the early visual system. The final input pattern consists of the normalized projection of the seven extracted blocks for the image on the top $n$ principal components.

## 4  Classifier design and training

The principal goal of classification for this study is to examine how the different representational spaces facilitate a classifiers ability to generalize to novel individuals. Comparing expected recognition rate error using the same classification technique with different representations should provide an indication of how well the signal of interest is preserved by the respective representation. A neural network with a hidden layer employing a non-linear activation function (sigmoid) is trained to learn the input-output mapping between the representation of the face image and the associated response vector given by human subjects.

A simple, fully connected, feed-forward neural network containing a single hidden layer with 10 nodes, when trained using back propagation, is capable of correctly classifying the input of training sets from each of the three representations (tested for pattern sizes up to 140 dimensions). The architecture of the network is fixed for a particular input size (based on the number of projections on the respective sub-space) and the generalization of the network is found on a set of images from a novel individual. An overview of the network design is shown in Fig. 2.

To minimize the impact of choosing a poor hold out set from the training set, each of the 11 individuals in the training set was in turn used as a hold out. The results of the 11 networks were then combined to evaluate the classification error on the test set. A number of different techniques are possible: winner take all, weighted average output, voting, etc. The method that we found to consistently give the highest generalization rate involved combining Z scores from the 11 networks. The average output for each possible emotion across all the networks was calculated along with its deviation over the entire training set. These values were used to normalize each output of the 11 networks and the highest weighted sum for a particular input was the associated emotion.

Due to the limited amount of data available for testing and training, a cross-validation technique using each set of an individual's images for testing was employed to increase the confidence of the generalization measurement. Thus, for each individual, 11 networks were combined to evaluate the generalization on the single test individual, and this procedure is repeated for all 12 individuals to give an average generalization error. This results in a total of 132 networks to evaluate the entire database. A single trial consisted of the generalization rate obtained over the whole database for a particular size of input pattern. By varying the initial weights of the network, we can determine the expected generalization performance of this type

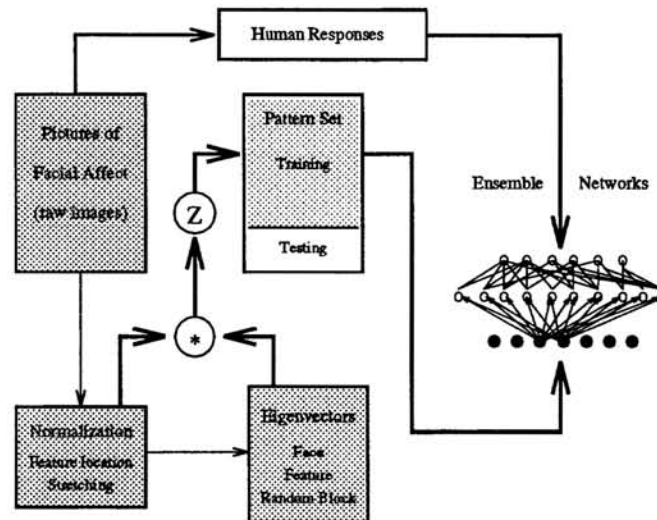

Figure 2: The processing path used to evaluate the pattern set of each representation scheme. The original image data (after normalization) is used to generate the eigenvectors and construct the pattern sets. Human responses are used in training the classifiers and determining generalization percentages on the test data.

classifier on each representation. The number of projections on the relevant space is also varied to determine a generalization curve for each representation. Constructing, training, and evaluating the 132 networks takes approximately 2 minutes on a SparcStation 10 for input pattern size of 15 and 4 minutes for an input pattern size of 80.

## 5   Results

Fig. 3 provides the expected generalization achieved by the neural network architecture initially seeded with small random weights for an increasing number of projections in the respective representational spaces. Each data point represents the average of 20 trials, 1 $\sigma$ error bars show the amount of error with respect to the mean. The curve (generalization rate vs. input pattern size) was evaluated at 6 points for the whole face and at 8 points for each feature based approach. The eigenfeature representation made use of up to 40 eigenvectors for the three regions while the random block representation made use of up to 17 eigenvectors for each of its seven regions.

For the most part, all the representations show improvement as the number of projections increase. Variations as input size increases are most likely due to a combination of two factors: lower signal to noise ratios (SNR) for higher order projections; and the increasing number of parameters with a fixed number of patterns, making generalization difficult. The highest average recognition rate achieved by the neural network ensembles is 86%, found using the random block representation with 15 projections per block. The results indicate that the generalization rate for emotion classification varies significantly depending on the representational strategy. Both local feature-based approaches (eigenfeatures and random block) did significantly better over their shared range than the eigenface representation. Over most of the range, the random block representation is clearly superior to the eigenfeature representation even though both are derived from the same image area.

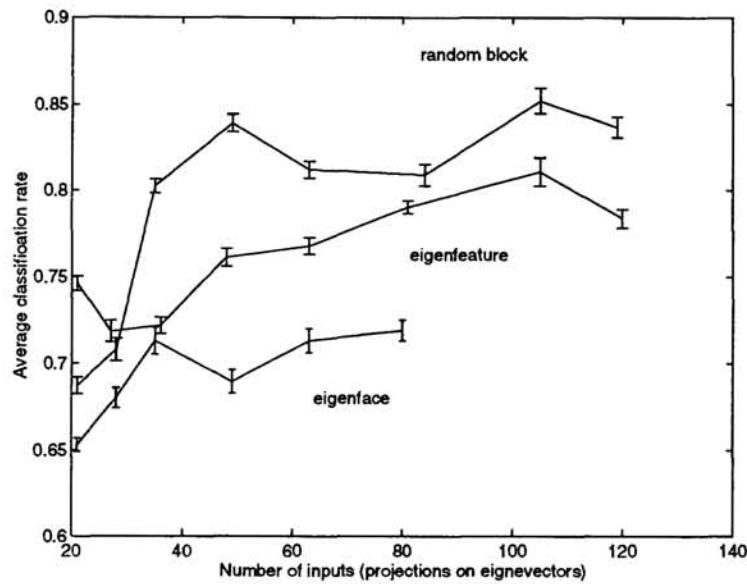

Figure 3: Generalization curves for feature-based representation and full-face representation.

## 6 Discussion

Fig. 3 clearly demonstrates that reasonable recognition rates can be obtained for novel individuals using representational techniques that were found useful for identity. The 86% generalization rate achieved by the neural network ensemble using random block patterns with 105 projections compares favorably with the results obtained from techniques that use an expression sequence (neutral to expression) [Mase, 1991, Yacoob and Davis, 1996, Bartlett et al., 1996]. Such schemes make use of a neutral mask which enhances the sequence's expression by simple subtraction, a technique that is not possible on novel, static face images. That our technique works as well or better indicates the possibility that the human visual system need not rely on difference image strategies over sequences of images in classifying emotions. As many psychological studies are performed on static images of individuals, models that can accommodate this aspect of emotion recognition can make predictions that directly guide research [Padgett et al., 1996].

As for the suitability of the various representations for fine grained discrimination over different individual objects (as required by emotion classification), Fig. 3 clearly demonstrates the benefits accrued by concentrating on facial features important to emotion. The generalization of the trained networks making use of the two local feature-based representations averages 6-15% higher than do the networks trained using projections on the eigenfaces. The increased performance can be attributed to a better signal to noise ratio for the feature regions. As much of the face is rigid (e.g. the chin and forehead), these regions provide little in the way of information useful in classifying emotions. However, there are substantial differences in these areas between individuals, which will be expressed by the principal component analysis of the images and thus reflected in the projected values. These variations are essentially noise with respect to emotion recognition making it more difficult for the classifier to extract useful generalizations during learning.

The final point is the superiority of the random block representation over the range

examined. One possible explanation for its significant performance edge is that major feature variations (e.g. open mouth, open eyes, etc.) are more effectively preserved by this representation than the eigenfeature approach, which covers the same image area. Due to individual differences in mouth/eye structure, one would expect that many of the eigenvectors of the feature space would be devoted to this variance. Facial expressions could be substantially orthogonal to this variance, so that information pertinent to emotion discrimination is effectively hidden. This of course would imply that the eigenfeature representation should be better than the random block representation for face recognition purposes. However, this is not the case. Nearest neighbor classification of individuals using the same pattern sets shows that the random block representation does better for this task as well (results not shown). We are currently developing a noise model that looks promising as an explanation for this phenomenon.

## 7  Conclusion

We have demonstrated that average generalization rates of 86% can be obtained for emotion recognition on novel individuals using techniques similar to work done in face recognition. Previous work on emotion recognition has relied on image sequences and obtained recognition rates of nearly the same generalization. The model we developed here is potentially of more interest to researchers in emotion that make use of static images of novel individuals in conducting their tests. Future work will compare aspects of the network model with human performance.

## References

[Bartlett et al., 1996] Bartlett, M., Viola, P., Sejnowski, T., Larsen, J., Hager, J., and Ekman, P. (1996). Classifying facial action. In Touretzky, D., Mozer, M., and Hasselmo, M., editors, *Advances in Neural Information Processing Systems 8*, Cambridge, MA. MIT Press.

[Brunelli and Poggio, 1993] Brunelli, R. and Poggio, T. (1993). Face recognition: Feature versus templates. *IEEE Trans. Patt. Anal. Machine Intell.*, 15(10).

[Cottrell and Metcalfe, 1991] Cottrell, G. W. and Metcalfe, J. (1991). Empath: Face, gender and emotion recognition using holons. In Lippman, R., Moody, J., and Touretzky, D., editors, *Advances in Neural Information Processing Systems 3*, pages 564–571, San Mateo. Morgan Kaufmann.

[Ekman and Friesen, 1976] Ekman, P. and Friesen, W. (1976). Pictures of facial affect.

[Ekman and Friesen, 1977] Ekman, P. and Friesen, W. (1977). *Facial Action Coding System*. Consulting Psychologists, Palo Alto, CA.

[Mase, 1991] Mase, K. (1991). Recognition of facial expression from optical flow. *IEICE Transactions*, 74(10):3474–3483.

[Padgett et al., 1996] Padgett, C., Cottrell, G., and Adolphs, R. (1996). Categorical perception in facial emotion classification. In Cottrell, G., editor, *Proceedings of the 18th Annual Cognitive Science Conference, San Diego CA*.

[Pentland et al., 1994] Pentland, A. P., Moghaddam, B., and Starner, T. (1994). View-based and modular eigenspaces for face recognition. In *IEEE Conference on Computer Vision & Pattern Recognition*.

[Yacoob and Davis, 1996] Yacoob, Y. and Davis, L. (1996). Recognizing human facial expressions from long image sequences using optical flow. *IEEE Transactions on Pattern Analysis and Machine Intelligence*, 18:636–642.
